# Kernel Logistic Regression and the Import Vector Machine

**Ji Zhu**
Department of Statistics
Stanford University
Stanford, CA 94305
*jzhu@stat.stanford.edu*

**Trevor Hastie**
Department of Statistics
Stanford University
Stanford, CA 94305
*hastie@stat.stanford.edu*

## Abstract

The support vector machine (SVM) is known for its good performance in binary classification, but its extension to multi-class classification is still an on-going research issue. In this paper, we propose a new approach for classification, called the import vector machine (IVM), which is built on kernel logistic regression (KLR). We show that the IVM not only performs as well as the SVM in binary classification, but also can naturally be generalized to the multi-class case. Furthermore, the IVM provides an estimate of the underlying probability. Similar to the "support points" of the SVM, the IVM model uses only a fraction of the training data to index kernel basis functions, typically a much smaller fraction than the SVM. This gives the IVM a computational advantage over the SVM, especially when the size of the training data set is large.

## 1 Introduction

In standard classification problems, we are given a set of training data $(x_1, y_1)$, $(x_2, y_2)$, $\ldots (x_N, y_N)$, where the output $y_i$ is qualitative and assumes values in a finite set $\mathcal{C}$. We wish to find a classfication rule from the training data, so that when given a new input $x$, we can assign a class $c$ from $\mathcal{C}$ to it. Usually it is assumed that the training data are an independently and identically distributed sample from an unknown probability distribution $P(X, Y)$.

The support vector machine (SVM) works well in binary classification, i.e. $y \in \{0, 1\}$, but its appropriate extension to the multi-class case is still an on-going research issue. Another weakness of the SVM is that it only estimates $sign[p(x) - 1/2]$, while the probability $p(x)$ is often of interest itself, where $p(x) = P(Y = 1 | X = x)$ is the conditional probability of a point being in class 1 given $X = x$. In this paper, we propose a new approach, called the import vector machine (IVM), to address the classification problem. We show that the IVM not only performs as well as the SVM in binary classification, but also can naturally be generalized to the multi-class case. Furthermore, the IVM provides an estimate of the probability $p(x)$. Similar to the "support points" of the SVM, the IVM model uses only a fraction of the training data to index the kernel basis functions. We call these training data *import points*. The computational cost of the SVM is $O(N^3)$, while the computational cost of the IVM is $O(N^2 q^2)$, where $q$ is the number of import points. Since $q$ does not tend to

increase as $N$ increases, the IVM can be faster than the SVM, especially for large training data sets. Empirical results show that the number of import points is usually much less than the number of support points.

In section (2), we briefly review some results of the SVM for binary classification and compare it with kernel logistic regression (KLR). In section (3), we propose our IVM algorithm. In section (4), we show some simulation results. In section (5), we generalize the IVM to the multi-class case.

## 2 Support vector machines and kernel logistic regression

The standard SVM produces a non-linear classification boundary in the original input space by constructing a linear boundary in a transformed version of the original input space. The dimension of the transformed space can be very large, even infinite in some cases. This seemingly prohibitive computation is achieved through a positive definite reproducing kernel $K$, which gives the inner product in the transformed space.

Many people have noted the relationship between the SVM and regularized function estimation in the reproducing kernel Hilbert spaces (RKHS). An overview can be found in Evgeniou *et al.* (1999), Hastie *et al.* (2001) and Wahba (1998). Fitting an SVM is equivalent to minimizing:

(1)
$$\frac{1}{N} \sum_{i=1}^{N} (1 - y_i f(x_i))_+ + \lambda \|f\|^2_{\mathcal{H}_K}.$$

with $f = b + h$, $h \in \mathcal{H}_K$, $b \in \mathcal{R}$. $\mathcal{H}_K$ is the RKHS generated by the kernel $K$. The classification rule is given by $sign[f]$.

By the representer theorem (Kimeldorf *et al* (1971)), the optimal $f(x)$ has the form:

(2)
$$f(x) = b + \sum_{i=1}^{N} a_i K(x, x_i).$$

It often happens that a sizeable fraction of the $N$ values of $a_i$ can be zero. This is a consequence of the truncation property of the first part of criterion (1). This seems to be an attractive property, because only the points on the wrong side of the classification boundary, and those on the right side but near the boundary have an influence in determining the position of the boundary, and hence have non-zero $a_i$'s. The corresponding $x_i$'s are called support points.

Notice that (1) has the form $loss + penalty$. The loss function $(1 - yf)_+$ is plotted in Figure 1, along with several traditional loss functions. As we can see, the negative log-likelihood (NLL) of the binomial distribution has a similar shape to that of the SVM. If we replace $(1 - yf)_+$ in (1) with $\ln(1 + e^{-yf})$, the NLL of the binomial distribution, the problem becomes a KLR problem. We expect that the fitted function performs similarly to the SVM for binary classfication.

There are two immediate advantages of making such a replacement: (a) Besides giving a classification rule, the KLR also offers a natural estimate of the probability $p(x) = e^f/(1 + e^f)$, while the SVM only estimates $sign[p(x) - 1/2]$; (b) The KLR can naturally be generalized to the multi-class case through kernel multi-logit regression, whereas this is not the case for the SVM. However, because the KLR compromises the hinge loss function of the SVM, it no longer has the "support points" property; in other words, all the $a_i$'s in (2) are non-zero.

KLR is a well studied problem; see Wahba *et al.* (1995) and references there; see also Green *et al.* (1985) and Hastie *et al.* (1990).

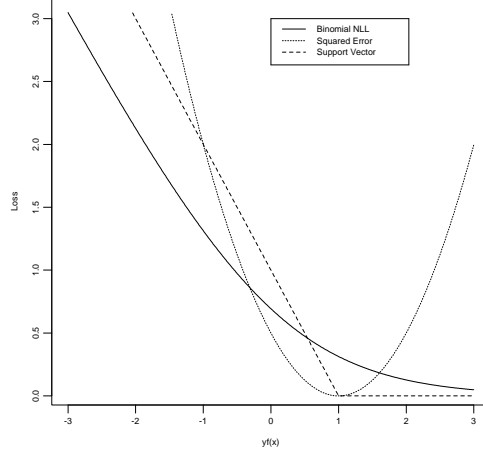

Figure 1: *Several loss functions, $y \in \{-1, 1\}$*

The computational cost of the KLR is $O(N^3)$; to save the computational cost, the IVM algorithm will find a sub-model to approximate the full model (2) given by the KLR. The sub-model has the form:

$$(3) \qquad f(x) = b + \sum_{x_i \in \mathcal{S}} a_i K(x, x_i)$$

where $\mathcal{S}$ is a subset of the training data $\{x_1, x_2, \ldots x_N\}$, and the data in $\mathcal{S}$ are called import points. The advantage of this sub-model is that the computational cost is reduced, especially for large training data sets, while not jeopardizing the performance in classification.

Several other researchers have investigated techniques in selecting the subset $\mathcal{S}$. Lin *et al.* (1998) divide the training data into several clusters, then randomly select a representative from each cluster to make up $\mathcal{S}$. Smola *et al.* (2000) develope a greedy technique to sequentially select $q$ columns of the kernel matrix $[K(x_i, x_j)]_{N \times N}$, such that the span of these $q$ columns approximates the span of $[K(x_i, x_j)]_{N \times N}$ well in the Frobenius norm. Williams *et al.* (2001) propose randomly selecting $q$ points of the training data, then using the Nystrom method to approximate the eigen-decomposition of the kernel matrix $[K(x_i, x_j)]_{N \times N}$, and expanding the results back up to $N$ dimensions. None of these methods uses the output $y_i$ in selecting the subset $\mathcal{S}$ (i.e., the procedure only involves $x_i$). The IVM algorithm uses both the output $y_i$ and the input $x_i$ to select the subset $\mathcal{S}$, in such a way that the resulting fit approximates the full model well.

## 3   Import vector machine

Following the tradition of logistic regression, we let $y_i \in \{0, 1\}$ for the rest of this paper. For notational simplicity, the constant term in the fitted function is ignored.

In the KLR, we want to minimize:

$$H = -\sum_{i=1}^{N} [y_i f(x_i) - \ln(1 + \exp(f(x_i)))] + \frac{\lambda}{2} \|f\|^2_{\mathcal{H}_K}$$

From (2), it can be shown that this is equivalent to the finite dimensional form:

$$(4) \qquad H = -\vec{y}^T (K_a \vec{a}) + \vec{1}^T \ln(1 + \exp(K_a \vec{a})) + \frac{\lambda}{2} \vec{a}^T K_q \vec{a}$$

where $\vec{a} = (a_1, \ldots a_N)^T$; the regressor matrix $K_a = [K(x_i, x_j)]_{N \times N}$; and the regularization matrix $K_q = K_a$.

To find $\vec{a}$, we set the derivative of $H$ with respect to $\vec{a}$ equal to 0, and use the Newton-Raphson method to iteratively solve the score equation. It can be shown that the Newton-Raphson step is a weighted least squares step:

(5) $$\vec{a}^{(k)} = (K_a^T W K_a + \lambda K_q)^{-1} K_a^T W \vec{z}$$

where $\vec{a}^{(k)}$ is the value of $\vec{a}$ in the $k$th step, $\vec{z} = (K_a \vec{a}^{(k-1)} + W^{-1}(\vec{y} - \vec{p}))$. The weight matrix is $W = diag[p(x_i)(1 - p(x_i))]_{N \times N}$.

As mentioned in section 2, we want to find a subset $\mathcal{S}$ of $\{x_1, x_2, \ldots x_N\}$, such that the sub-model (3) is a good approximation of the full model (2). Since it is impossible to search for every subset $\mathcal{S}$, we use the following greedy forward strategy:

## 3.1 Basic algorithm

(B1) Let $\mathcal{S} = \emptyset$, $\mathcal{R} = \{x_1, x_2, \ldots, x_N\}$, $k = 1$.

(B2) For each $x_l \in \mathcal{R}$, let

$$f_l(x) = \sum_{x_j \in \mathcal{S} \cup \{x_l\}} a_j K(x, x_j)$$

Find $\vec{a}$ to minimize

$$H(x_l) \quad = \quad -\sum_{i=1}^{N}[y_i f_l(x_i) - \ln(1 + \exp(f_l(x_i)))] + \frac{\lambda}{2}\|f_l(x)\|_{\mathcal{H}_K}^2$$

(6) $$= \quad -\vec{y}^T(K_a^l \vec{a^l}) + \vec{1}^T \ln(1 + \exp(K_a^l \vec{a^l})) + \frac{\lambda}{2}\vec{a^l}^T K_q^l \vec{a^l}$$

where the regressor matrix $K_a^l = [K(x_i, x_j)]_{N \times (q+1)}$, $x_i \in \{x_1, x_2, \ldots x_N\}$, $x_j \in \mathcal{S} \cup \{x_l\}$; the regularization matrix $K_q^l = [K(x_j, x_l)]_{(q+1) \times (q+1)}$, $x_j, x_l \in \mathcal{S} \cup \{x_l\}$; $q = |\mathcal{S}|$.

(B3) Let

$$x_{l^*} = \operatorname{argmin}_{x_l \in \mathcal{R}} H(x_l).$$

Let $\mathcal{S} = \mathcal{S} \cup \{x_{l^*}\}$, $\mathcal{R} = \mathcal{R} \setminus \{x_{l^*}\}$, $H_k = H(x_{l^*})$, $k = k + 1$.

(B4) Repeat steps (B2) and (B3) until $H_k$ converges.

We call the points in $\mathcal{S}$ import points.

## 3.2 Revised algorithm

The above algorithm is computationally feasible, but in step (B2) we need to use the Newton-Raphson method to find $\vec{a}$ iteratively. When the number of import points $q$ becomes large, the Newton-Raphson computation can be expensive. To reduce this computation, we use a further approximation.

Instead of iteratively computing $\vec{a}^{(k)}$ until it converges, we can just do a one-step iteration, and use it as an approximation to the converged one. To get a good approximation, we take advantage of the fitted result from the current "optimal" $\mathcal{S}$, i.e., the sub-model when $|\mathcal{S}| = q$, and use it as the initial value. This one-step update is similar to the score test in generalized linear models (GLM); but the latter does not have a penalty term. The updating formula allows the weighted regression (5) to be computed in $O(Nq)$ time.

Hence, we have the revised step (B2) for the basic algorithm:

($B2^*$) For each $x_l \in \mathcal{R}$, correspondingly augment $K_a$ with a column, and $K_q$ with a column and a row. Use the updating formula to find $\vec{a}$ in (5). Compute (6).

### 3.3 Stopping rule for adding point to $\mathcal{S}$

In step ($B4$) of the basic algorithm, we need to decide whether $H_k$ has converged. A natural stopping rule is to look at the regularized NLL. Let $H_1, H_2, \ldots$ be the sequence of regularized NLL's obtained in step ($B4$). At each step $k$, we compare $H_k$ with $H_{k-r}$, where $r$ is a pre-chosen small integer, for example $r = 1$. If the ratio $\frac{|H_k - H_{k-r}|}{|H_k|}$ is less than some pre-chosen small number $\alpha$, for example, $\alpha = 0.001$, we stop adding new import points to $\mathcal{S}$.

### 3.4 Choosing the regularization paramter $\lambda$

So far, we have assumed that the regularization parameter $\lambda$ is fixed. In practice, we also need to choose an "optimal" $\lambda$. We can randomly split all the data into a training set and a tuning set, and use the misclassification error on the tuning set as a criterion for choosing $\lambda$. To reduce the computation, we take advantage of the fact that the regularized NLL converges faster for a larger $\lambda$. Thus, instead of running the entire revised algorithm for each $\lambda$, we propose the following procedure, which combines both adding import points to $\mathcal{S}$ and choosing the optimal $\lambda$:

($C1$) Start with a large regularization parameter $\lambda$.

($C2$) Let $\mathcal{S} = \emptyset$, $\mathcal{R} = \{x_1, x_2, \ldots, x_N\}$, $k = 1$.

($C3$) Run steps ($B2^*$), ($B3$) and ($B4$) of the revised algorithm, until the stopping criterion is satisfied at $\mathcal{S} = \{x_{i1}, \ldots, x_{iq_k}\}$ . Along the way, also compute the misclassfication error on the tuning set.

($C4$) Decrease $\lambda$ to a smaller value.

($C5$) Repeat steps ($C3$) and ($C4$), starting with $\mathcal{S} = \{x_{i1}, \ldots, x_{iq_k}\}$.

We choose the optimal $\lambda$ as the one that corresponds to the minimum misclassification error on the tuning set.

## 4 Simulation

In this section, we use a simulation to illustrate the IVM method. The data in each class are generated from a mixture of Gaussians (Hastie *et al.* (2001)). The simulation results are shown in Figure 2.

### 4.1 Remarks

The support points of the SVM are those which are close to the classification boundary or misclassified and usually have large weights $[p(x)(1 - p(x))]$. The import points of the IVM are those that decrease the regularized NLL the most, and can be either close to or far from the classification boundary. This difference is natural, because the SVM is only concerned with the classification $sign[p(x) - 1/2]$, while the IVM also focuses on the unknown probability $p(x)$. Though points away from the classification boundary do not contribute to determining the position of the classification boundary, they may contribute to estimating the unknown probability $p(x)$. Figure 3 shows a comparison of the SVM and the IVM. The total computational cost of the SVM is $O(N^3)$, while the computational cost of the IVM method is $O(N^2 q^2)$, where $q$ is the number of import points. Since $q$ does not

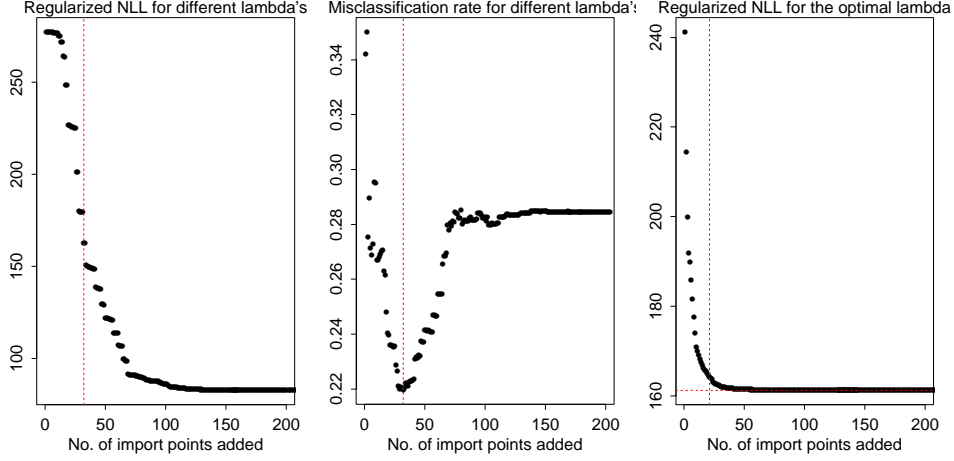

Figure 2: *Radial kernel is used. $N = 200$. The left and middle panels illustrate how to choose the optimal $\lambda$. $r = 1$, $\alpha = 0.001$, $\lambda$ decreases from $e^{10}$ to $e^{-10}$. The minimum misclassification rate $0.219$ is found to correspond to $\lambda = 0.135$. The right panel is for the optimal $\lambda = 0.135$. The stopping criterion is satisfied when $|\mathcal{S}| = 21$.*

tend to increase as $N$ increases, the computational cost of the IVM can be smaller than that of the SVM, especially for large training data sets.

## 5   Multi-class case

In this section, we briefly describe a generalization of the IVM to multi-class classification. Suppose there are $M + 1$ classes. We can write the response as an $M$-vector $\vec{y}$, with each component being either 0 or 1, indicating which class the observation is in. Therefore $y_k = 1$, $y_j = 0$, $j \neq k$, $j \leq M$ indicates the response is in the $k$th class, and $y_j = 0, j \leq M$ indicates the response is in the $M + 1$th class. Using the $M + 1$th class as the basis, the multi-logit can be written as $f_1 = \ln(p_1/p_{M+1})$, ..., $f_M = \ln(p_M/p_{M+1})$, $f_{M+1} = 0$. Hence the Bayes classification rule is given by:

$$c = \operatorname{argmax}_{k \in \{1,2,\ldots,M+1\}} f_k$$

We use $i$ to index the observations, $j$ to index the classes, i.e. $i = 1, \ldots N$, $j = 1, \ldots M$. Then the regularized negative log-likelihood is

$$(7) \qquad H = - \sum_{i=1}^{N} [\vec{y}_i^T \vec{f}(x_i) - \ln(1 + e^{f_1(x_i)} + \cdots + e^{f_M(x_i)})] + \frac{\lambda}{2} \|f\|_{\mathcal{H}_K}^2$$

where $\vec{y}_i = (y_{i1}, y_{i2}, \ldots, y_{iM})^T$, $\vec{f}(x_i) = (f_1(x_i), f_2(x_i), \ldots, f_M(x_i))^T$, and

$$\|f\|_{\mathcal{H}_K}^2 = \sum_{j=1}^{M} \|f_j\|_{\mathcal{H}_K}^2$$

Using the representer theorem (Kimeldorf *et al.* (1971)), the $j$th element of $\vec{f}(x)$, $f_j(x)$, which minimizes $H$ has the form

$$(8) \qquad f_j(x) = \sum_{i=1}^{N} a_{ij} K(x, x_i).$$

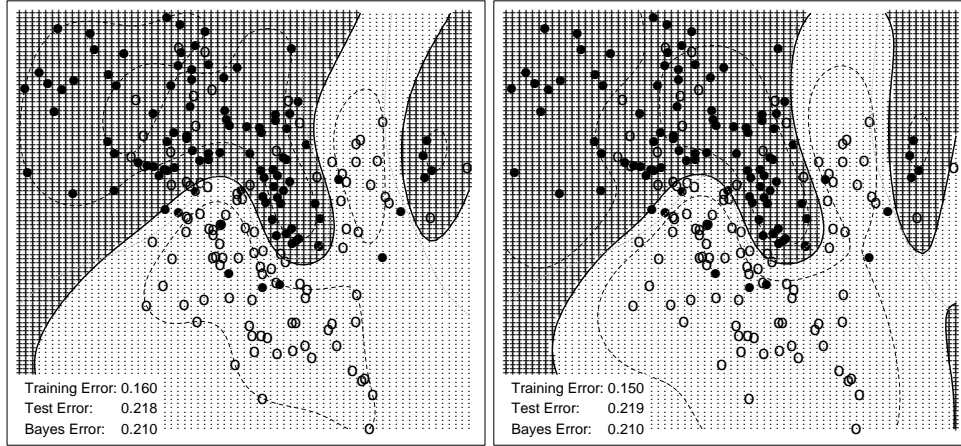

SVM - with 107 support points          IVM - with 21 import points

Training Error: 0.160
Test Error:  0.218
Bayes Error:  0.210

Training Error: 0.150
Test Error:  0.219
Bayes Error:  0.210

Figure 3: *The solid lines are the classification boundaries; the dotted lines are the Bayes rule boundaries. For the SVM, the dashed lines are the edges of the margin. For the IVM, the dashed lines are the $p(x) = 0.25$ and $0.75$ lines.*

Hence, (7) becomes

$$(9) \qquad H = -\sum_{i=1}^{N}[\vec{y}_i^{T}(K_a(i,)A)^{T} - \ln(1 + \vec{1}^{T}e^{(K_a(i,)A)^{T}})] + \frac{\lambda}{2}\sum_{j=1}^{M}\vec{a}_j^{T}K_q\vec{a}_j$$

where $A = (\vec{a}_1 \ldots \vec{a}_M) = (a_{ij})$, $K_a$ and $K_q$ are defined in the same way as in the binary case; and $K_a(i,)$ is the $i$th row of $K_a$.

The multi-class IVM procedure is similar to the binary case, and the computational cost is $O(MN^2q^2)$. Figure 4 is a simulation of the multi-class IVM. The data in each class are generated from a mixture of Gaussians (Hastie *et al.* (2001)).

Multi-class IVM - with 32 import points

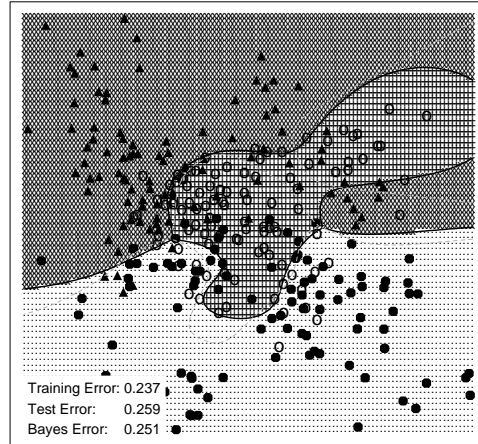

Training Error: 0.237
Test Error:  0.259
Bayes Error:  0.251

Figure 4: *Radial kernel is used. $M + 1 = 3$, $N = 300$, $\lambda = 0.368$, $|\mathcal{S}| = 32$.*

# 6  Conclusion

We have discussed the import vector machine (IVM) method in both binary and multi-class classification. We showed that it not only performs as well as the SVM, but also provides an estimate of the probability $p(x)$. The computational cost of the IVM is $O(N^2q^2)$ for the binary case and $O(MN^2q^2)$ for the multi-class case, where $q$ is the number of import points.

## Acknowledgments

We thank Dylan Small, John Storey, Rob Tibshirani, and Jingming Yan for their helpful comments. Ji Zhu is partially supported by the Stanford Graduate Fellowship. Trevor Hastie is partially supported by grant DMS-9803645 from the National Science Foundation, and grant ROI-CA-72028-01 from the National Institutes of Health. Thanks to Grace Wahba and Chris Williams for pointing out several interesting and important references. We also want to thank the anonymous NIPS referees who helped improve this paper.

## References

[1] Burges, C.J.C. (1998) A tutorial on support vector machines for pattern recognition. In *Data Mining and Knowledge Discovery.* Kluwer Academic Publishers, Boston. (Volume 2)

[2] Evgeniou, T., Pontil, M., & Poggio., T. (1999) Regularization networks and support vector machines. In A.J. Smola, P. Bartlett, B. Schölkopf, and C. Schuurmans, editors, *Advances in Large Margin Classifiers.* MIT Press.

[3] Green, P. & Yandell, B. (1985) Semi-parametric generalized linear models. *Proceedings 2nd International GLIM Conference*, Lancaster, Lecture notes in Statistics No. 32 44-55 Springer-Verlag, New York.

[4] Hastie, T. & Tibshirani, R. (1990) *Generalized Additive Models*, Chapman and Hall.

[5] Hastie, T., Tibshirani, R., & Friedman, J.(2001) *The elements of statistical learning.* In print.

[6] Lin, X., Wahba, G., Xiang, D., Gao, F., Klein, R. & Klein B. (1998), Smoothing spline ANOVA models for large data sets with Bernoulli observations and the randomized GACV. Technical Report 998, Department of Statistics, University of Wisconsin, Madison WI.

[7] Kimeldorf, G. & Wahba, G. (1971) Some results on Tchebycheffian spline functions. *J. Math. Anal. Applic.* **33**, 82-95.

[8] Smola, A. & Schölkopf, B. (2000) Sparse Greedy Matrix Approximation for Machine Learning. In *Proceedings of the Seventeenth International Conference on Machine Learning.* Morgan Kaufmann Publishers.

[9] Wahba, G. (1998) Support Vector Machine, Reproducing Kernel Hilbert Spaces and the Randomized GACV. Technical Report 984rr, Department of Statistics, University of Wisconsin, Madison WI.

[10] Wahba, G., Gu, C., Wang, Y., & Chappell, R. (1995) Soft Classification, a.k.a. Risk Estimation, via Penalized Log Likelihood and Smoothing Spline Analysis of Variance. In D.H. Wolpert, editor, *The Mathematics of Generalization.* Santa Fe Institute Studies in the Sciences of Complexity. Addison-Wesley Publisher.

[11] Williams, C. & Seeger, M (2001) Using the Nystrom Method to Speed Up Kernel Machines. In T. K. Leen, T. G. Diettrich, and V. Tresp, editors, *Advances in Neural Information Processing Systems 13.* MIT Press.
